# Approximate inference algorithms for two-layer Bayesian networks

**Andrew Y. Ng**
Computer Science Division
UC Berkeley
Berkeley, CA 94720
*ang@cs.berkeley.edu*

**Michael I. Jordan**
Computer Science Division and
Department of Statistics
UC Berkeley
Berkeley, CA 94720
*jordan@cs.berkeley.edu*

## Abstract

We present a class of approximate inference algorithms for graphical models of the QMR-DT type. We give convergence rates for these algorithms and for the Jaakkola and Jordan (1999) algorithm, and verify these theoretical predictions empirically. We also present empirical results on the difficult QMR-DT network problem, obtaining performance of the new algorithms roughly comparable to the Jaakkola and Jordan algorithm.

## 1 Introduction

The graphical models formalism provides an appealing framework for the design and analysis of network-based learning and inference systems. The formalism endows graphs with a joint probability distribution and interprets most queries of interest as marginal or conditional probabilities under this joint. For a fixed model one is generally interested in the conditional probability of an output given an input (for prediction), or an input conditional on the output (for diagnosis or control). During learning the focus is usually on the likelihood (a marginal probability), on the conditional probability of unobserved nodes given observed nodes (e.g., for an EM or gradient-based algorithm), or on the conditional probability of the parameters given the observed data (in a Bayesian setting).

In all of these cases the key computational operation is that of marginalization. There are several methods available for computing marginal probabilities in graphical models, most of which involve some form of message-passing on the graph. Exact methods, while viable in many interesting cases (involving sparse graphs), are infeasible in the dense graphs that we consider in the current paper. A number of approximation methods have evolved to treat such cases; these include search-based methods, loopy propagation, stochastic sampling, and variational methods.

Variational methods, the focus of the current paper, have been applied successfully to a number of large-scale inference problems. In particular, Jaakkola and Jordan (1999) developed a variational inference method for the QMR-DT network, a benchmark network involving over 4,000 nodes (see below). The variational method provided accurate approximation to posterior probabilities within a second of computer time. For this difficult

inference problem exact methods are entirely infeasible (see below), loopy propagation does not converge to correct posteriors (Murphy, Weiss, & Jordan, 1999), and stochastic sampling methods are slow and unreliable (Jaakkola & Jordan, 1999).

A significant step forward in the understanding of variational inference was made by Kearns and Saul (1998), who used large deviation techniques to analyze the convergence rate of a simplified variational inference algorithm. Imposing conditions on the magnitude of the weights in the network, they established a $O(\sqrt{\log N/N})$ rate of convergence for the error of their algorithm, where $N$ is the fan-in.

In the current paper we utilize techniques similar to those of Kearns and Saul to derive a new set of variational inference algorithms with rates that are faster than $O(\sqrt{\log N/N})$. Our techniques also allow us to analyze the convergence rate of the Jaakkola and Jordan (1999) algorithm. We test these algorithms on an idealized problem and verify that our analysis correctly predicts their rates of convergence. We then apply these algorithms to the difficult the QMR-DT network problem.

## 2   Background

### 2.1   The QMR-DT network

The QMR-DT (Quick Medical Reference, Decision-Theoretic) network is a bipartite graph with approximately 600 top-level nodes $d_i$ representing diseases and approximately 4000 lower-level nodes $f_j$ representing findings (observed symptoms). All nodes are binary-valued. Each disease is given a prior probability $P(d_i = 1)$, obtained from archival data, and each finding is parameterized as a "noisy-OR" model:

$$P(f_i = 1|d) = 1 - e^{-\theta_{i0} - \sum_{j \in \pi_i} \theta_{ij} d_j},$$

where $\pi_i$ is the set of parent diseases for finding $f_i$, and where the parameters $\theta_{ij}$ are obtained from assessments by medical experts (see Shwe, et al., 1991).

Letting $z_i = \theta_{i0} + \sum_{j \in \pi_i} \theta_{ij} d_j$, we have the following expression for the likelihood[1]:

$$P(f) = \sum_{\{d\}} \left[ \prod_{i=1}^{K} \left(1 - e^{-z_i}\right)^{f_i} \prod_{i=1}^{K} \left(e^{-z_i}\right)^{1-f_i} \prod_{j=1}^{N} P(d_j) \right], \qquad (1)$$

where the sum is a sum across the approximately $2^{600}$ configurations of the diseases. Note that the second product, a product over the negative findings, factorizes across the diseases $d_j$; these factors can be absorbed into the priors $P(d_j)$ and have no significant effect on the complexity of inference. It is the positive findings which couple the diseases and prevent the sum from being distributed across the product.

Generic exact algorithms such as the junction tree algorithm scale exponentially in the size of the maximal clique in a moralized, triangulated graph. Jaakkola and Jordan (1999) found cliques of more than 150 nodes in QMR-DT; this rules out the junction tree algorithm. Heckerman (1989) discovered a factorization specific to QMR-DT that reduces the complexity substantially; however the resulting algorithm still scales exponentially in the number of positive findings and is only feasible for a small subset of the benchmark cases.

## 2.2 The Jaakkola and Jordan (JJ) algorithm

Jaakkola and Jordan (1999) proposed a variational algorithm for approximate inference in the QMR-DT setting. Briefly, their approach is to make use of the following variational inequality:

$$1 - e^{-z_i} \le e^{\lambda_i z_i - c_i},$$

where $c_i$ is a deterministic function of $\lambda_i$. This inequality holds for arbitrary values of the free "variational parameter" $\lambda_i$. Substituting these variational upper bounds for the probabilities of positive findings in Eq. (1), one obtains a factorizable upper bound on the likelihood. Because of the factorizability, the sum across diseases can be distributed across the joint probability, yielding a product of sums rather than a sum of products. One then minimizes the resulting expression with respect to the variational parameters to obtain the tightest possible variational bound.

## 2.3 The Kearns and Saul (KS) algorithm

A simplified variational algorithm was proposed by Kearns and Saul (1998), whose main goal was the theoretical analysis of the rates of convergence for variational algorithms. In their approach, the local conditional probability for the finding $f_i$ is approximated by its value at a point a small distance $\varepsilon_i$ above or below (depending on whether upper or lower bounds are desired) the mean input $E[z_i]$. This yields a variational algorithm in which the values $\varepsilon_i$ are the variational parameters to be optimized. Under the assumption that the weights $\theta_{ij}$ are bounded in magnitude by $\tau/N$, where $\tau$ is a constant and $N$ is the number of parent ("disease") nodes, Kearns and Saul showed that the error in likelihood for their algorithm converges at a rate of $O(\sqrt{\log N/N})$.

# 3 Algorithms based on local expansions

Inspired by Kearns and Saul (1998), we describe the design of approximation algorithms for QMR-DT obtained by expansions around the mean input to the finding nodes. Rather than using point approximations as in the Kearns-Saul (KS) algorithm, we make use of Taylor expansions. (See also Plefka (1982), and Barber and van de Laar (1999) for other perturbational techniques.)

Consider a generalized QMR-DT architecture in which the noisy-OR model is replaced by a general function $\psi(z) : \mathbb{R} \to [0, 1]$ having uniformly bounded derivatives, i.e., $|\psi^{(i)}(z)| \le B_i$. Define $F(z_1, \ldots, z_K) = \prod_{i=1}^{K} (\psi(z_i))^{f_i} \prod_{i=1}^{K} (1 - \psi(z_i))^{1-f_i}$ so that the likelihood can be written as

$$P(f) = E_{\{z_i\}}[F(z_1, \ldots, z_K)]. \tag{2}$$

Also define $\mu_i = E[z_i] = \theta_{i0} + \sum_{j=1}^{N} \theta_{ij} P(d_j = 1)$.

A simple mean-field-like approximation can be obtained by evaluating $F$ at the mean values $\mu_i$:

$$P(f) \approx F(\mu_1, \ldots, \mu_K). \tag{3}$$

We refer to this approximation as "MF(0)."

Expanding the function $F$ to second order, and defining $\epsilon_i = z_i - \mu_i$, we have:

$$
P(f) = E_{\{\epsilon_i\}} \left[ F(\vec{\mu}) + \sum_{i_1=1}^{K} F_{i_1}(\vec{\mu})\epsilon_{i_1} + \frac{1}{2!} \sum_{i_1=1}^{K} \sum_{i_2=1}^{K} F_{i_1 i_2}(\vec{\mu})\epsilon_{i_1}\epsilon_{i_2} + \right.
$$
$$
\left. \frac{1}{3!} \sum_{i_1=1}^{K} \sum_{i_2=1}^{K} \sum_{i_3=1}^{K} F_{i_1 i_2 i_3}(\vec{\xi}_\epsilon)\epsilon_{i_1}\epsilon_{i_2}\epsilon_{i_3} \right] \tag{4}
$$

where the subscripts on $F$ represent derivatives. Dropping the remainder term and bringing the expectation inside, we have the "MF(2)" approximation:

$$P(f) \approx F(\vec{\mu}) + \frac{1}{2} \sum_{i_1=1}^{K} \sum_{i_2=1}^{K} F_{i_1 i_2}(\vec{\mu}) \mathrm{E}[\epsilon_{i_1} \epsilon_{i_2}]$$

More generally, we obtain a "MF($i$)" approximation by carrying out a Taylor expansion to $i$-th order.

### 3.1 Analysis

In this section, we give two theorems establishing convergence rates for the MF($i$) family of algorithms and for the Jaakkola and Jordan algorithm. As in Kearns and Saul (1998), our results are obtained under the assumption that the weights are of magnitude at most $O(1/N)$ (recall that $N$ is the number of disease nodes). For large $N$, this assumption of "weak interactions" implies that each $z_i$ will be close to its mean value with high probability (by the law of large numbers), and thereby gives justification to the use of local expansions for the probabilities of the findings.

Due to space constraints, the detailed proofs of the theorems given in this section are deferred to the long version of this paper, and we will instead only sketch the intuitions for the proofs here.

**Theorem 1** *Let $K$ (the number of findings) be fixed, and suppose $|\theta_{ij}| \leq \frac{\tau}{N}$ for all $i, j$ for some fixed constant $\tau$. Then the absolute error of the MF($k$) approximation is $O\left(\frac{1}{N^{(k+1)/2}}\right)$ for $k$ odd and $O\left(\frac{1}{N^{(k/2+1)}}\right)$ for $k$ even.*

**Proof intuition.** First consider the case of odd $k$. Since $|\theta_{ij}| \leq \frac{\tau}{N}$, the quantity $\epsilon_i = z_i - \mu_i = \sum_j \theta_{ij}(d_j - \mathrm{E}[d_j])$ is like an average of $N$ random variables, and hence has standard deviation on the order $1/\sqrt{N}$. Since MF($k$) matches $F$ up to the $k$-th order derivatives, we find that when we take a Taylor expansion of MF($k$)'s error, the leading non-zero term is the $k + 1$-st order term, which contains quantities such as $\epsilon_i^{k+1}$. Now because $\epsilon_i$ has standard deviation on the order $1/\sqrt{N}$, it is unsurprising that $\mathrm{E}[\epsilon_i^{k+1}]$ is on the order $1/N^{(k+1)/2}$, which gives the error of MF($k$) for odd $k$.

For $k$ even, the leading non-zero term in the Taylor expansion of the error is a $k + 1$-st order term with quantities such as $\epsilon_i^{k+1}$. But if we think of $\epsilon_i$ as converging (via a central limit theorem effect) to a symmetric distribution, then since symmetric distributions have small odd central moments, $\mathrm{E}[\epsilon_i^{k+1}]$ would be small. This means that for $k$ even, we may look to the order $k + 2$ term for the error, which leads to MF($k$) having the the same big-$O$ error as MF($k + 1$). Note this is also consistent with how MF(0) and MF(1) always give the same estimates and hence have the same absolute error.                                                            $\square$

A theorem may also be proved for the convergence rate of the Jaakkola and Jordan (JJ) algorithm. For simplicity, we state it here only for noisy-OR networks.[2] A closely related result also holds for sigmoid networks with suitably modified assumptions; see the full paper.

**Theorem 2** *Let $K$ be fixed, and suppose $\psi(z) = 1 - e^{-z}$ is the noisy-OR function. Suppose further that $0 \leq \theta_{ij} \leq \frac{\tau}{N}$ for all $i, j$ for some fixed constant $\tau$, and that $\mu_i \geq \mu_{\min}$ for all $i$, for some fixed $\mu_{\min} > 0$. Then the absolute error of the JJ approximation is $O\left(\frac{1}{N}\right)$.*

The condition of some $\mu_{\min}$ lowerbounding the $\mu_i$'s ensures that the findings are not too unlikely; for it to hold, it is sufficient that there be bias ("leak") nodes in the network with weights bounded away from zero.

**Proof intuition.** Neglecting negative findings, (which as discussed do not need to be handled variationally,) this result is proved for a "simplified" version of the JJ algorithm, that always chooses the variational parameters so that for each $i$, the exponential upperbound on $\psi(z_i)$ is tangent to $\psi$ at $z_i = \mu_i$. (The "normal" version of JJ can have error no worse than this simplified one.) Taking a Taylor expansion again of the approximation's error, we find that since the upperbound has matched zeroth and first derivatives with $F$, the error is a second order term with quantities such as $\epsilon_i^2$. As discussed in the MF($k$) proof outline, this quantity has expectation on the order $1/N$, and hence JJ's error is $O(1/N)$. □

To summarize our results in the most useful cases, we find that MF(0) has a convergence rate of $O(1/N)$, both MF(2) and MF(3) have rates of $O(1/N^2)$, and JJ has a convergence rate of $O(1/N)$.

## 4 Simulation results

### 4.1 Artificial networks

We carried out a set of simulations that were intended to verify the theoretical results presented in the previous section. We used bipartite noisy-OR networks, with full connectivity between layers and with the weights $\theta_{ij}$ chosen uniformly in $(0, 2/N)$. The number $N$ of top-level ("disease") nodes ranged from 10 to 1000. Priors on the disease nodes were chosen uniformly in $(0, 1)$.

The results are shown in Figure 1 for one and five positive findings (similar results where obtained for additional positive findings).

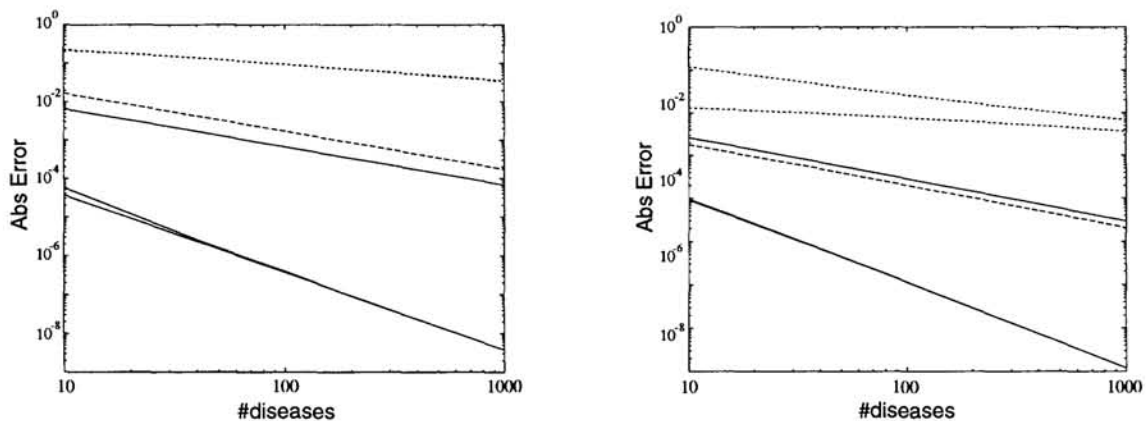

Figure 1: Absolute error in likelihood (averaged over many randomly generated networks) as a function of the number of disease nodes for various algorithms. The short-dashed lines are the KS upper and lower bounds (these curves overlap in the left panel), the long-dashed line is the JJ algorithm and the solid lines are MF(0), MF(2) and MF(3) (the latter two curves overlap in the right panel).

The results are entirely consistent with the theoretical analysis, showing nearly exactly the expected slopes of -1/2, -1 and -2 on a loglog plot.[3] Moreover, the asymptotic results are

also predictive of overall performance: the MF(2) and MF(3) algorithms perform best in all cases, MF(0) and JJ are roughly equivalent, and KS is the least accurate.

## 4.2 QMR-DT network

We now present results for the QMR-DT network, in particular for the four benchmark CPC cases studied by Jaakkola and Jordan (1999). These cases all have fewer than 20 positive findings; thus it is possible to run the Heckerman (1989) "Quickscore" algorithm to obtain the true likelihood.

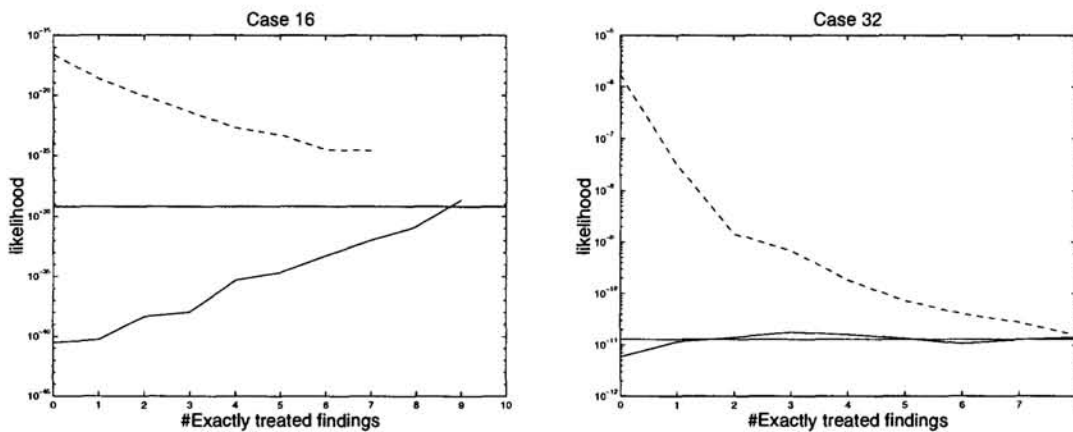

Figure 2: Results for CPC cases 16 and 32, for different numbers of exactly treated findings. The horizontal line is the true likelihood, the dashed line is JJ's estimate, and the lower solid line is MF(3)'s estimate.

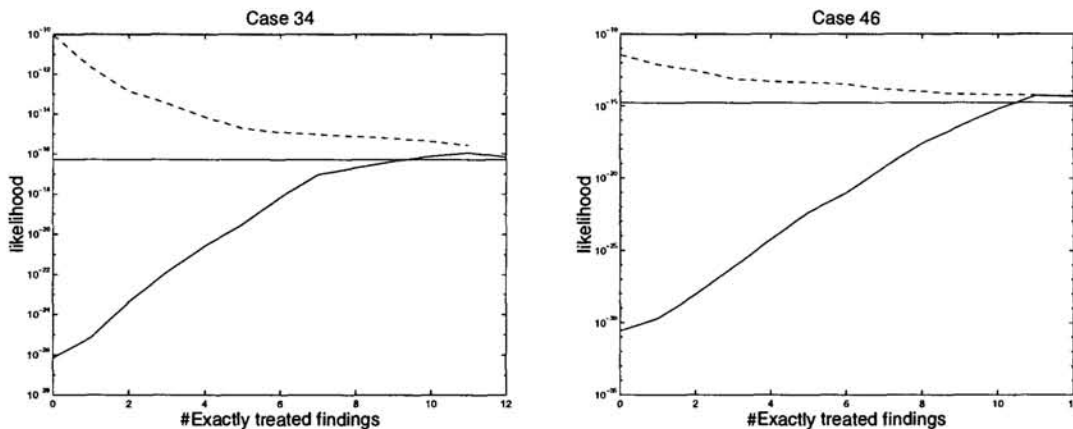

Figure 3: Results for CPC cases 34 and 46. Same legend as above.

In Jaakkola and Jordan (1999), a hybrid methodology was proposed in which only a portion of the findings were treated approximately; exact methods were used to treat the remaining findings. Using this hybrid methodology, Figures 2 and 3 show the results of running JJ and MF(3) on these four cases.[4]

The results show the MF algorithm yielding results that are comparable with the JJ algorithm.

## 5 Conclusions and extension to multilayer networks

This paper has presented a class of approximate inference algorithms for graphical models of the QMR-DT type, supplied a theoretical analysis of convergence rates, verified the rates empirically, and presented promising empirical results for the difficult QMR-DT problem.

Although the focus of this paper has been two-layer networks, the MF($k$) family of algorithms can also be extended to multilayer networks. For example, consider a 3-layer network with nodes $b_i$ being parents of nodes $d_i$ being parents of nodes $f_i$. To approximate $\Pr[f]$ using (say) MF(2), we first write $\Pr[f]$ as an expectation of a function ($F$) of the $z_i$'s, and approximate this function via a second-order Taylor expansion. To calculate the expectation of the Taylor approximation, we need to calculate terms in the expansion such as $\mathrm{E}[d_i]$, $\mathrm{E}[d_i d_j]$ and $\mathrm{E}[d_i^2]$. When $d_i$ had no parents, these quantities were easily derived in terms of the disease prior probabilities. Now, they instead depend on the joint distribution of $d_i$ and $d_j$, which we use our two-layer version of MF($k$), applied to the first two ($b_i$ and $d_i$) layers of the network, to approximate. It is important future work to carefully study the performance of this algorithm in the multilayer setting.

### Acknowledgments

We wish to acknowledge the helpful advice of Tommi Jaakkola, Michael Kearns, Kevin Murphy, and Larry Saul.

## Footnotes

[1]In this expression, the factors $P(d_j)$ are the probabilities associated with the (parent-less) disease nodes, the factors $(1 - e^{-z_i})$ are the probabilities of the (child) finding nodes that are observed to be in their positive state, and the factors $e^{-z_i}$ are the probabilities of the negative findings. The resulting product is the joint probability $P(f, d)$, which is marginalized to obtain the likelihood $P(f)$.

[2]Note in any case that JJ can be applied only when $\psi$ is log-concave, such as in noisy-OR networks (where incidentally all weights are non-negative).

[3]The anomalous behavior of the KS lower bound in the second panel is due to the fact that the algorithm generally finds a vacuous lower bound of 0 in this case, which yields an error which is essentially constant as a function of the number of diseases.

[4]These experiments were run using a version of the JJ algorithm that optimizes the variational parameters just once without any findings treated exactly, and then uses these fixed values of the parameters thereafter. The order in which findings are chosen to be treated exactly is based on JJ's estimates, as described in Jaakkola and Jordan (1999). Missing points in the graphs for cases 16 and

### References

[1] Barber, D., & van de Laar, P. (1999) Variational cumulant expansions for intractable distributions. *Journal of Artificial Intelligence Research, 10*, 435–455.

[2] Heckerman, D. (1989). A tractable inference algorithm for diagnosing multiple diseases. In *Proceedings of the Fifth Conference on Uncertainty in Artificial Intelligence*.

[3] Jaakkola, T. S., & Jordan, M. I. (1999). Variational probabilistic inference and the QMR-DT network. *Journal of Artificial Intelligence Research, 10*, 291–322.

[4] Jordan, M. I., Ghahramani, Z., Jaakkola, T. S., & Saul, L. K. (1998). An introduction to variational methods for graphical models. In *Learning in Graphical Models*. Cambridge: MIT Press.

[5] Kearns, M. J., & Saul, L. K. (1998). Large deviation methods for approximate probabilistic inference, with rates of convergence. In G. F. Cooper & S. Moral (Eds.), *Proceedings of the Fourteenth Conference on Uncertainty in Artificial Intelligence*. San Mateo, CA: Morgan Kaufmann.

[6] Murphy, K. P., Weiss, Y., & Jordan, M. I. (1999). Loopy belief propagation for approximate inference: An empirical study. In *Proceedings of the Fifteenth Conference on Uncertainty in Artificial Intelligence*.

[7] Plefka, T. (1982). Convergence condition of the TAP equation for the infinite-ranged Ising spin glass model. In *J. Phys. A: Math. Gen., 15*(6).

[8] Shwe, M., Middleton, B., Heckerman, D., Henrion, M., Horvitz, E., Lehmann, H., & Cooper, G. (1991). Probabilistic diagnosis using a reformulation of the INTERNIST-1/QMR knowledge base I. The probabilistic model and inference algorithms. *Methods of Information in Medicine, 30*, 241–255.

---

34 correspond to runs where our implementation of the Quickscore algorithm encountered numerical problems.